# Implementation of Neural Hardware with the Neural VLSI of URAN in Applications with Reduced Representations

Il-Song Han
Korea Telecom Research Laboratories
17, Woomyun-dong, Suhcho-ku
Seoul 137-140, KOREA

Ki-Chul Kim
Dept. of Info and Comm
KAIST
Seoul, 130-012, Korea

Hwang-Soo Lee
Dept. of Info and Comm
KAIST
Seoul, 130-012, Korea

## Abstract

This paper describes a way of neural hardware implementation with the analog-digital mixed mode neural chip. The full custom neural VLSI of Universally Reconstructible Artificial Neural network(URAN) is used to implement Korean speech recognition system. A multi-layer perceptron with linear neurons is trained successfully under the limited accuracy in computations. The network with a large frame input layer is tested to recognize spoken korean words at a forward retrieval. Multichip hardware module is suggested with eight chips or more for the extended performance and capacity.

# 1 INTRODUCTION

In general, the neural network hardware or VLSI has been preferred in respects of its relatively fast speed, huge network size and effective cost comparing to software simulation. Universally Reconstructible Artificial Neural-network(URAN), the new analog-digital mixed VLSI neural network, can be used for the implementation of the real world neural network applications with digital interface. The basic electronic synapse circuit is based on the electrically controlled MOSFET resistance and is operated with discrete pulses.

The URAN's adaptability is tested for the multi-layer perceptron with the reduced precision of connections and states. The linear neuron function is also designed for the real world applications. The multi-layer network with back propagation learning is designed for the speaker independent digit/word recognition. The other case of application is for the servo control, where the neural input and output are extended to 360 levels for the suitable angle control. With the servo control simulation, the flexibility of URAN is proved to extend the accuracy of input and output from external.

## 2. Analog-Digital Mixed Chip - URAN

In the past, there have been improvements in analog or analog-digital mixed VLSI chips. Analog neural chips or analog-digital mixed neural chips are still suffered from the lack of accuracy, speed or flexibility. With the proposed analog-digital mixed neural network circuit of URAN, the accuracy is improved by using the voltage-controlled linear MOSFET resistance for the synapse weight emulation. The speed in neural computation is also improved by using the simple switch controlled by the neural input as described in previous works.

The general flexibility is attained by the independent characteristic of each synapse cell and the modular structure of URAN chip. As in Table 1 of URAN chip feature, the chip is operated under the flexible control, that is, the various mode of synaptic connection per neuron or the extendable weight accuracy can be implemented. It is not limited for the asynchronous/direct interchip expansion in size or speed. In fact, 16 fully connected module of URAN is selected from external and independently - it is possible to select either one by one or all at once.

Table 1. URAN Chip Features

| | |
|---|---|
| Total Synapses | 135,424 connections |
| Computation Speed | 200 Giga Connections Per S |
| Weight Accuracy | 8 Bit |
| Module No. | 16 |
| Module Size | 92 X 92 |

As all circuits over the chip except digital decoder unit are operated in analog transistor level, the computation speed is relatively high and even can be improved substantially. The cell size including interconnection area in conventional short-channel technology is reduced less than 900 $\mu$ m². From its expected and measured linear characteristic, URAN has the accuracy more than 256 linear levels.

The accuracy extendability and flexible modularity are inherent in electrical wired-OR characteristics as each synapse is an independent bipolar current source with switch. No additional clocking or any limited synchronous operation is required in this case, while it is indispensible in most of conventional digital neural hardware or analog-digital neural chip. Therefore, any size of neural network can be integrated in VLSI or module hardware merely by placing the cell in 2 dimensional array without any timing limitation or loading effect.

## 3. Neural Hardware with URAN – Module Expansion

URAN is the full custom VLSI of analog-digital mixed operation. The prototype of URAN chip is fabricated in $1.0\mu$ digital CMOS technology. The chip contains 135,424 synapses with 8 bit weight accuracy on a 13 X 13 mm² die size using single poly double metal technology. As summarized in Table 1 of chip features, the chip allows the variety of configuration. In the prototype chip, 16 fully connected module of 92 X 92 can be selected from external and independently – selecting independent module either one by one, several or all at a time is possible.

With URAN's synapse circuit of linear voltage-controlled bipolar current source, the synaptic multiplication with weight value is done with the switching transistor, in a similar way of analog-sampled data type. The accuracy enhancement and flexible modularity of URAN are inherent in its electrical wired-OR interface from each independent bipolar current source. And the neural network hardware module can be realized in any size with the multi URAN chips.

## 4. Considerations on the Reduced Precision

URAN chip is applied for the case of Korean speaker independent speech recognition. By changing numbers of hidden units and input accuracy, the result of simulations has not shown any problems in recognition accuracy. It means that the overall performance is not severely affected from the accuracy of weight, input, and output with URAN. Also, it was possible to train with 2 or 1 decimal accuracy for input and output, which is equivalent to 8 bit or 4 bit precisions. With 20 hidden units for the Korean spoken 10 digit recognition, 2 decimal input accuracy yields 99.2% and 1 decimal input accuracy yields 98.6%,

while binary 1-bit input results 96.6%.

The following is the condition for the experimentation. The general result is summarized in Table 2.

*Conditions for Training and Test*
■ 2,000 samples from 10 women and 10 men
  ( 10 times X 10 digits X 20 persons )
□ Training with 500 spoken samples of 10 digits in Korean from 10 persons
  (5 times X 10 digits X [ 5 women and 5 men ] ) from 2,000 samples
□ Recognition Test with 1,000 spoken samples from
  the other 10 persons of women and men.
*Preprocessing of samples*
□ sampled at 10KHz with 12bit accuracy
□ preemphasis with 0.95
□ Hamming window of 20ms
□ 17 channel critical-band filter bank
□ noise added for the SNR of 30dB, 20dB, 10dB, 0dB

### Table 2.  Low Accuracy Connection   with Linear Neuron

| SNR Ratio | Input / Output Accuracy | | |
|---|---|---|---|
| | 2 decimal | 1 decimal | 1 bit |
| clean | 97.5% | 97.2% | 90.7% |
| 30 dB | 96.2% | 96.6% | 90.5% |
| 20 dB | 90.1% | 91.3% | 86.6% |
| 10 dB | 59.8% | 59.9% | 68.0% |
| 0 dB | 30.8% | 29.5% | 38.5% |

In case of servo control, the digital VCR for industrial purpose is modelled for the application.  Six inputs are used to minimize the number of hidden units and 20 hidden units are configured for one output.  For the adaptation to URAN, the linear neuron function is used during the simulation. The weight accuracy during the learning phase using conventional computer is 4 byte and that in the recall phase using URAN chip is 1 byte. With this limitation, the overall performance is not severely degraded, that is, the reduction of error is attained up to 70% improvement comparing to the conventional method.  The nonideal factor of 30% results from the limitation in learning data as well as the limited hardware. Current results are suitable for the digital VCR or compact camcoder in noisy environment

## 5. Conclusion

In this paper, it is proved to be suitable for the application to the multi-layer perceptron with the use of URAN chip, which is fabricated in conventional digital CMOS technology - $1.0\mu$ single poly double metal. The reduced weight accuracy of 1 byte is proved to be enough to obtain high performance using the linear neuron and URAN.

With 8 test chips of 135,424 connections, it is now under development of the practical module of neural hardware with million connections and tera connections per second - comparable to the power of biological neuro-system of some insects. The size of the hardware is smaller than A4 size and is designed for more general recognition system. The flexible modularity of URAN makes it possible to realize a 1,000,000 connections neural chip in $0.5\mu$ CMOS technology and a general purpose neural hardware of hundreds of tera connections or more.

## References

Il Song Han and Ki-Hwan Ahn, "Neural Network VLSI Chip Implementation of Analog-Digital Mixed Operation for more than 100,000 Connections" MicroNeuro'93, pp. 159-162, 1993

M. Brownlow, L. Tarassenko, A. F. Murray, A. Hamilton, I S Han, H. M. Reekie, "Pulse Firing Neural Chips Implementing Hundreds of Neurons," NIPS2, pp. 785-792, 1990

